# On Strategy Stitching in Large Extensive Form Multiplayer Games

**Richard Gibson and Duane Szafron**
Department of Computing Science, University of Alberta
Edmonton, Alberta, T6G 2E8, Canada
{rggibson | dszafron}@ualberta.ca

## Abstract

Computing a good strategy in a large extensive form game often demands an extraordinary amount of computer memory, necessitating the use of abstraction to reduce the game size. Typically, strategies from abstract games perform better in the real game as the granularity of abstraction is increased. This paper investigates two techniques for stitching a base strategy in a coarse abstraction of the full game tree, to expert strategies in fine abstractions of smaller subtrees. We provide a general framework for creating static experts, an approach that generalizes some previous strategy stitching efforts. In addition, we show that static experts can create strong agents for both 2-player and 3-player Leduc and Limit Texas Hold'em poker, and that a specific class of static experts can be preferred among a number of alternatives. Furthermore, we describe a poker agent that used static experts and won the 3-player events of the 2010 Annual Computer Poker Competition.

## 1 Introduction

Many sequential decision-making problems are commonly modelled as an *extensive form game*. Extensive games are very versatile due to their ability to represent multiple agents, imperfect information, and stochastic events.

For many real-world problems, however, the extensive form game representation is too large to be feasibly handled by current techniques. To address this limitation, strategies are often computed in abstract versions of the game that group similar states together into single abstract states. For very large games, these abstractions need to be quite coarse, leaving many different states indistinguishable. However, for smaller subtrees of the full game, strategies can be computed in much finer abstractions. Such "expert" strategies can then be pieced together, typically connecting to a "base strategy" computed in the full coarsely-abstracted game. A disadvantage of this approach is that we may make assumptions about the other agents' strategies. In addition, by computing the base strategy and the experts separately, we may lose "cohesion" among the different components.

We investigate stitched strategies in extensive form games, focusing on the trade-offs between the sizes of the abstractions versus the assumptions made and the cohesion among the computed strategies. We define two strategy stitching techniques: (i) *static experts* that are computed in very fine abstractions with varying degrees of assumptions and little cohesion, and (ii) *dynamic experts* that are contained in abstractions with lower granularity, but make fewer assumptions and have perfect cohesion. This paper generalizes previous strategy stitching efforts [1, 2, 11] under a more general static expert framework. We use poker as a testbed to demonstrate that, despite recent mixed results, static experts can create much stronger overall agents than the base strategy alone. Furthermore, we show that under a fixed memory limitation, a specific class of static experts are preferred to several

alternatives. As a final validation of these results, we describe entries to the 2010 Annual Computer Poker Competition[1] (ACPC) that used static experts to win the 3-player events.

## 2   Background

An extensive form game [9] is a rooted directed tree, where nodes represent decision states, edges represent actions, and terminal nodes hold end-game utility values for players. For each player, the decision states are partitioned into information sets such that game states within an information set are indistinguishable to the player. Non-singleton information sets arise due to hidden information that is only available to a subset of the players, such as private cards in poker. More formally:

**Definition 2.1 (Osborne and Rubenstein [9, p. 200])** *A finite **extensive game** $\Gamma$ with imperfect information has the following components:*

- *A finite set $N$ of **players**.*
- *A finite set $H$ of sequences, the possible **histories** of actions, such that the empty sequence is in $H$ and every prefix of a sequence in $H$ is also in $H$. $Z \subseteq H$ are the terminal histories (those which are not a prefix of any other sequence). $A(h) = \{a \mid ha \in H\}$ are the actions available after a nonterminal history $h \in H$.*
- *A function $P$ that assigns to each nonterminal history $h \in H \backslash Z$ a member of $N \cup \{C\}$. $P$ is the **player function**. $P(h)$ is the player who takes an action after the history $h$. If $P(h) = C$, then chance determines the action taken after history $h$. Define $H_i := \{h \in H \mid P(h) = i\}$.*
- *A function $f_C$ that associates with every history $h$ for which $P(h) = C$ a probability measure $f_C(\cdot|h)$ on $A(h)$ ($f_C(a|h)$ is the probability that $a$ occurs given $h$), where each such probability measure is independent of every other such measure.*
- *For each player $i \in N$ a partition $\mathcal{I}_i$ of $H_i$ with the property that $A(h) = A(h')$ whenever $h$ and $h'$ are in the same member of the partition. For $I \in \mathcal{I}_i$, we denote by $A(I)$ the set of $A(h)$ and by $P(I)$ the player $P(h)$ for any $h \in I$. $\mathcal{I}_i$ is the **information partition** of player $i$; a set $I \in \mathcal{I}_i$ is an **information set** of player $i$.*
- *For each player $i \in N$ a utility function $u_i$ from the terminal histories $Z$ to the real numbers $\mathbf{R}$. If $N = \{1, 2\}$ and $u_1 = -u_2$, it is a **2-player zero-sum extensive game**. Define $\Delta_{u,i} := \max_z u_i(z) - \min_z u_i(z)$ to be the range of the utilities for player $i$.*

A **strategy for player** $i$, $\sigma_i$, is a function such that for each information set $I \in \mathcal{I}_i$, $\sigma_i(I)$ is a probability distribution over $A(I)$. Let $\Sigma_i$ be the set of all strategies for player $i$. For $h \in I$, we define $\sigma_i(h) := \sigma_i(I)$. A **strategy profile** $\sigma$ consists of a strategy $\sigma_i$ for each player $i \in N$. We let $\sigma_{-i}$ refer to all the strategies in $\sigma$ except $\sigma_i$, and denote $u_i(\sigma)$ to be the expected utility for player $i$ given that all players play according to $\sigma$.

In a 2-player zero-sum game, a **best response** to a player 1 strategy $\sigma_1$ is a player 2 strategy $\sigma_2^{\text{BR}} = \text{argmax}_{\sigma_2} u_2(\sigma_1, \sigma_2)$ (similarly for a player 2 strategy $\sigma_2$). The best response value of $\sigma_1$ is $u_2(\sigma_1, \sigma_2^{\text{BR}})$, which measures the **exploitability** of $\sigma_1$. The exploitability of a strategy tells us how much that strategy loses to a worst-case opponent. Outside of 2-player zero-sum games, the worst-case scenario for player $i$ would be for all other players to minimize player $i$'s utility instead of maximizing their own. In large games, this value is difficult to compute since opponents cannot share private information. Thus, we only investigate exploitability for 2-player zero-sum games.

Counterfactual regret minimization (CFR) [14] is an iterative procedure for computing strategy profiles in extensive form games. In 2-player zero-sum games, CFR produces an approximate Nash equilibrium profile. In addition, CFR strategies have also been found to compete very well in games with more than 2 players [1]. CFR's memory requirements are proportional to the number of information sets in the game times the number of actions available at an information set.

The extensive form game representation of many real-world problems is too large to feasibly compute a strategy directly. A common approach in these games is to first create an *abstract game* by combining information sets into single abstract states or by disallowing certain actions:

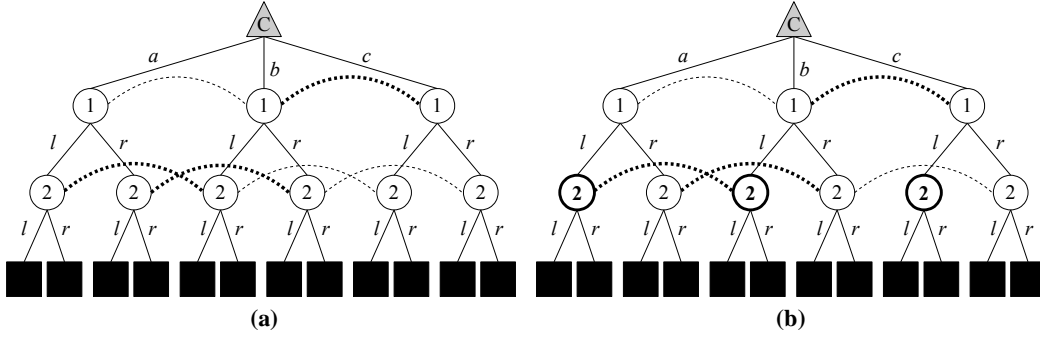

**Figure 1: (a)** An abstraction of an extensive game, where states connected by a bold curve are in the same information set and thin curves denote merged abstract information sets. In the unabstracted game, player 1 cannot distinguish between whether chance generated $b$ or $c$ and player 2 cannot distinguish between $a$ and $b$. In the abstract game, neither player can distinguish between any of chance's outcomes. **(b)** An example of a game $\Gamma'$ derived from the unabstracted game $\Gamma$ in (a) for a dynamic expert strategy. Here, the abstraction from (a) is used as the base abstraction, and the null abstraction is employed on the subtree with $G_{1,1} = \emptyset$ and $G_{2,1} = \{al, bl, cl\}$ (bold states).

**Definition 2.2 (Waugh *et al.* [12])** *An **abstraction for player i** is a pair $\alpha_i = \left\langle \alpha_i^{\mathcal{I}}, \alpha_i^A \right\rangle$, where*

- *$\alpha_i^{\mathcal{I}}$ is a partition of $H_i$ defining a set of abstract information sets coarser than $\mathcal{I}_i$ (i.e., every $I \in \mathcal{I}_i$ is a subset of some set in $\alpha_i^{\mathcal{I}}$), and*
- *$\alpha_i^A$ is a function on histories where $\alpha_i^A(h) \subseteq A(h)$ and $\alpha_i^A(h) = \alpha_i^A(h')$ for all histories $h$ and $h'$ in the same abstract information set. We will call this the abstract action set.*

*The **null abstraction** for player $i$ is $\phi_i = \langle \mathcal{I}_i, A \rangle$. An **abstraction** $\alpha$ is a set of abstractions $\alpha_i$, one for each player. Finally, for any abstraction $\alpha$, the **abstract game**, $\Gamma^\alpha$, is the extensive game obtained from $\Gamma$ by replacing $\mathcal{I}_i$ with $\alpha_i^{\mathcal{I}}$ and $A(h)$ with $\alpha_i^A(h)$ when $P(h) = i$, for all $i \in N$.*

Figure 1a shows an example of an abstracted extensive form game with no action abstraction. By reducing the number of information sets, computing strategies in an abstract game with an algorithm such as CFR requires less memory than computing strategies in the real game. Intuitively, if a strategy profile for the abstract game $\sigma$ performs well in $\Gamma^\alpha$, and if $\alpha_i^{\mathcal{I}}$ is defined such that merged information sets are "strategically similar," then $\sigma$ is also likely to perform well in $\Gamma$. Identifying strategically similar information sets can be delicate though and typically becomes a domain-specific task. Nevertheless, we often would like to have as much granularity in our abstraction as will fit in memory to allow computed strategies to be as diverse as necessary.

## 3 Strategy Stitching

To achieve abstractions with finer granularity, a natural approach is to break the game up into subtrees, abstract each of the subtrees, and compute a strategy for each abstract subtree independently. We introduce a formalism for doing so that generalizes Waugh *et al.*'s strategy grafting [11] and two poker-specific methods described in Section 5. First, select a subset $S \subseteq N$ of players. Secondly, for each $i \in S$, compute a base strategy $\sigma_i$ for playing the full game. Next, divide the game into subtrees:

**Definition 3.1 (Waugh *et al.* [11])** $G_i = \{G_{i,0}, G_{i,1}, ..., G_{i,p}\}$ *is a **grafting partition for player i** if*

- *$G_i$ is a partition of $H_i$ (possibly containing empty parts),*
- *$\forall I \in \mathcal{I}_i, \exists j \in \{0, 1, ..., p\}$ such that $I \subseteq G_{i,j}$, and*
- *$\forall j \in \{1, 2, ..., p\}, h \in G_{i,j},$ and $h' \in H_i,$ if $h$ is a prefix of $h',$ then $h' \in G_{i,j} \cup G_{i,0}.$*

For each $i \in S$, choose a grafting partition $G_i$ so that each partition has an equal number of parts $p$. Then, compute a strategy, or *static expert*, for each subtree using any strategy computation technique, such as CFR. Finally, since the subtrees are disjoint, create a *static expert strategy* by combining the static experts without any overlap to the base strategy in the undivided game:

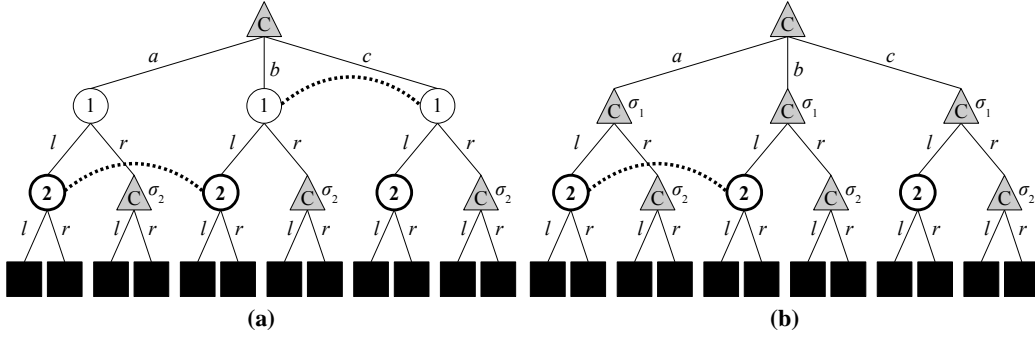

**Figure 2:** Two examples of a game $\Gamma^j$ for a static expert derived from the unabstracted game $\Gamma$ in Figure 1a. In both (a) and (b), $G_{2,j} = \{al, bl, cl\}$ (bold states). If player 1 takes action $r$, player 2 no longer controls his or her decisions. Player 2's actions are instead generated by the base strategy $\sigma_2$, computed beforehand. In (a), we have $S = \{2\}$. On the other hand, in (b), $S = N = \{1, 2\}$, $G_{1,j} = \emptyset$, and hence all of player 1's actions are seeded by the base strategy $\sigma_1$.

**Definition 3.2** *Let $S \subseteq N$ be a nonempty subset of players. For each $i \in S$, let $\sigma_i$ be a strategy for player $i$ and $G_i = \{G_{i,0}, G_{i,1}, ..., G_{i,p}\}$ be a grafting partition for player $i$. For $j \in \{1, 2, ..., p\}$, define $\Gamma^j$ to be an extensive game derived from the original game $\Gamma$ where, for all $i \in S$ and $h \in H_i \backslash G_{i,j}$, we set $P(h) = C$ and $f_C(a|h) = \sigma_i(h, a)$. That is, each player $i \in S$ only controls actions for histories in $G_{i,j}$ and is forced to play according to $\sigma_i$ elsewhere. Let the **static expert** of $\{G_{i,j} \mid i \in S\}$, $\sigma^j$, be a strategy profile of the game $\Gamma^j$. Finally, define the **static expert strategy for player i**, $\sigma_i^S$, as*

$$\sigma_i^S(h, a) := \left\{ \begin{array}{ll} \sigma_i(h, a) & \text{if } h \in G_{i,0} \\ \sigma_i^j(h, a) & \text{if } h \in G_{i,j}. \end{array} \right.$$

*We call $\{\sigma_i \mid i \in S\}$ the **base** or **seeding strategies** and $\{G_i \mid i \in S\}$ the **grafting profile** for the static expert strategy $\sigma_i^S$.*

Figure 2 shows two examples of a game $\Gamma^j$ for a single static expert. This may be the only subtree for which a static expert is computed ($p = 1$), or there could be more subtrees contained in the grafting partition(s) ($p > 1$). Under a fixed memory limitation, we can employ finer abstractions for the subtrees $\Gamma^j$ than we can in the full game $\Gamma$. This is because $\Gamma^j$ removes some of the information sets belonging to players in $S$, freeing up memory for computing strategies on the subtrees.

When $|S| = 1$, the static expert approach is identical to strategy grafting [11, Definition 8], with the exception that each static expert need not be an approximate Nash equilibrium. We relax the definition for static experts because Nash equilibria are difficult to compute in multiplayer games, and may not be the best solution concept outside of 2-player zero-sum games anyways. Choosing $|S| > 1$, however, is dangerous because we fix opponent probabilities and assume that our opponents are "static" at certain locations. For example, in Figure 2b, it may not be wise for player 2 to assume that player 1 must follow $\sigma_1$. Doing so can dramatically skew player 2's beliefs about the action generated by chance and hurt the expert's performance against opponents that do not follow $\sigma_1$. As we will see in Section 6, having more static experts with $|S| > 1$ can result in a more exploitable static expert strategy. On the other hand, by removing information sets for multiple players, the static expert approach creates smaller subtrees than strategy grafting does. As a result, we can employ even finer abstractions within the subtrees. Section 6 shows that despite the risks, the abstraction gains often lead to static experts with $S = N$ being preferred.

Regardless of the choice of $S$, the base strategy lacks "cohesion" with the static experts since its computation is based on its own play at the subtrees rather than the experts' play. Though the experts are identically seeded, the base strategy may want to play towards the expert subtrees more often to increase utility. This observation motivates our introduction of *dynamic experts* that are computed concurrently with a base. The full extensive game is divided into subtrees and each subtree is supplied its own abstraction:

**Definition 3.3** *Let $\alpha^0, \alpha^1, ..., \alpha^p$ be abstractions for the game $\Gamma$ and for each $i \in N$, let $G_i = \{G_{i,0}, G_{i,1}, ..., G_{i,p}\}$ be a grafting partition for player $i$ satisfying $I \cap G_{i,j} \in \{\emptyset, I\}$ for all $j \in \{0, ..., p\}$ and $I \in \alpha_i^{j,\mathcal{I}}$. Thus, each abstract information set is contained entirely in some part of the grafting partition. Let $\Gamma'$ be the abstract game obtained from $\Gamma$ by replacing $\mathcal{I}_i$ with $\bigcup_{j=0}^{p}\{I \in \alpha_i^{j,\mathcal{I}} \mid I \subseteq G_{i,j}\}$ and $A(h)$ with $\alpha_i^{j,A}(h)$ when $P(h) = i$ and $h \in G_{i,j}$, for all $i \in N$. Let the **dynamic expert strategy for player i**, $\sigma_i'$, be a strategy for player $i$ of the game $\Gamma'$. Finally define the **dynamic expert** of $G_{i,j}$, $\sigma_i^j$, to be $\sigma_i'$ restricted to the histories in $G_{i,j}$, $\sigma_i'|_{G_{i,j}}$. The abstraction $\alpha^0$ is denoted as the **base abstraction** and the dynamic expert $\sigma_i^0$ is denoted as the **base strategy**.*

Figure 1b contains an abstract game tree $\Gamma'$ for a dynamic expert strategy. We can view a dynamic expert strategy as a strategy computed in an abstraction with differing granularity dependent on the history of actions taken. Note that our definition is somewhat redundant to the definition of abstraction as we are simply defining a new abstraction for $\Gamma$ based on the abstractions $\alpha^0, \alpha^1, ..., \alpha^p$. Nonetheless, we supply Definition 3.3 to provide the terms in bold that we will use throughout.

Under memory constraints, a dynamic expert strategy typically sacrifices abstraction granularity in the base strategy to achieve finer granularity in the experts. We hope doing so achieves better performance at parts of the game that we believe may be more important. For instance, importance could depend on the predicted relative frequencies of reaching different subtrees. The base strategy's abstraction is reduced to guarantee perfect cohesion between the base and the experts; the base strategy knows about the experts and can calculate its probabilities "dynamically" during strategy computation based on the feedback from the experts. In Section 6, we contrast static and dynamic experts to compare this trade-off between abstraction size and strategy cohesion.

## 4 Texas and Leduc Hold'em

A hand of Texas Hold'em poker (or simply Hold'em) begins with each player being dealt two private cards, and two players posting mandatory bets or *blinds*. There are four betting rounds, the *pre-flop*, *flop*, *turn*, and *river* where five community cards are successively revealed. Of the players that did not fold, the player with the highest ranked poker hand wins all of the bets. Full rules can be found on-line.[2] We focus on the Limit Hold'em variant that fixes the bet sizes and the number of bets allowed per round. We denote the players' actions as $f$ (*fold*), $c$ (*check* or *call*), and $r$ (*bet* or *raise*).

Leduc Hold'em [10] (or simply Leduc) is a smaller version of Hold'em, played with a six card deck consisting of two Jacks, two Queens, and two Kings with only two betting rounds, pre-flop and flop. Rather than using blinds, antes are posted by all players at the beginning of a hand. Only one private card is dealt to each player and one community card is dealt on the flop.

While Leduc is small enough to bypass abstraction, Hold'em is a massive game in terms of the number of information sets; 2-player Limit Hold'em has approximately $3 \times 10^{14}$ information sets, and 3-player has roughly $5 \times 10^{17}$. Applying CFR to these enormous state spaces necessitates abstraction. A common abstraction technique in poker is to group many different card dealings into single abstract states or *buckets*. This is commonly done by ordering all possible poker hands for a specific betting round according to some metric, such as expected hand strength ($\mathbf{E}[\text{HS}]$) or expected hand strength squared ($\mathbf{E}[\text{HS}^2]$), and then grouping hands with similar metric values into the same bucket [7]. *Percentile bucketing* with $N$ buckets and $M$ hands puts the top $M/N$ hands into 1 bucket, the next best $M/N$ into a second bucket, etc., so that the buckets are approximately equal in size. More advanced bucketing schemes that use multiple metrics and clustering techniques are possible, but our experiments use simple percentile bucketing with no action abstraction.

## 5 Related Work

Our general framework for applying static experts to any extensive form game captures some previous poker-specific strategy stitching approaches. First, the PsOpti family of agents [2], which play 2-player Limit Hold'em, contain a base strategy called the "pre-flop model" and 7 static experts with $S = N$, or "post-flop models." Due to resource and technology limitations, the abstractions used to

build the pre-flop and post-flop models were quite coarse, making the family no match for today's top agents. Secondly, Abou Risk and Szafron [1] attach 6 static experts with $S = N$ (which they call "heads-up experts") to a base strategy for playing 3-player Limit Hold'em. Each expert focuses on a subtree immediately following a fold action, allowing much finer abstractions for these 2-player scenarios. However, their results were mixed as the stitched strategy was not always better than the base strategy alone. Nonetheless, our positive results for static experts with $S = N$ in Section 6 provide evidence that the PsOpti approach and heads-up experts are indeed credible.

In addition, Gilpin and Sandholm [5] create a poker agent for 2-player Limit Hold'em that uses a 2-phase strategy different from the approaches discussed thus far. The first phase is used to play the pre-flop and flop rounds, and is computed similarly to the PsOpti pre-flop model. For the turn and river rounds, a second phase strategy is computed on-line. One drawback of this approach is that the on-line computations must be quick enough to play in real time. Despite fixing the flop cards, this constraint forced the authors to still employ a very coarse abstraction during the second phase.

Furthermore, there have been a few other related approaches to creating poker agents. While 2-player poker is well studied, Ganzfried and Sandholm [3, 4] developed algorithms for computing Nash equilibria in multiplayer games and applied it to a small 3-player jam/fold poker game. Additionally, Gilpin et al. [6] use an automated abstraction building tool to dynamically bucket hands in 2-player Limit Hold'em. Here, we are not concerned with equilibrium properties or the abstraction building process itself. In fact, strategy stitching is orthogonal to both strategy computation and abstraction improvements, and could be used in conjunction with more sophisticated techniques.

## 6 Empirical Evaluation

In this section, we create several stitched strategies in both Leduc and Hold'em using the chance-sampled variant of CFR [14]. CFR is state of the art in terms of memory efficiency for strategy computation, allowing us to employ abstractions with higher granularity than otherwise possible. Results may differ with other techniques for computing strategies and building abstractions. While CFR requires iterations quadratic in the number of information sets to converge [14, Theorem 4], we restrict our resources only in terms of memory. Even though Leduc is small enough to not necessitate strategy stitching, the Leduc experiments were conducted to evaluate our hypothesis that static experts with $S = N$ can improve play. We ran many experiments and for brevity, only a representative sample of the results are summarized.

To be consistent with post-flop models [2] and heads-up experts [1], our grafting profiles are defined only in terms of the players' actions. For each history $h \in H$, define $b := b(h)$ to be the subsequence of $h$ obtained by removing all actions generated by chance. We refer to a *b-expert for player* $i$ as an expert constructed for the subtree $G_i(b) := \{h \in H_i \mid b \text{ is a prefix of } b(h)\}$ containing all histories where the players initially follow $b$. For example, the experts for the games in Figures 1b, 2a, and 2b are $l$-experts because the game is split after player 1 takes action $l$.

**Leduc.** Our Leduc experiments use three different base abstractions, one of which is simply the null abstraction. The second and third abstractions are the "JQ-K" and "J-QK" abstractions that, on the pre-flop, cannot distinguish between whether the private card is a Jack or Queen, or whether the private card is a Queen or King respectively. In addition, these two abstractions can only distinguish between whether the flop card pairs with the private card or not rather than knowing the identity of the flop card. Because Leduc is such a small game, we do not consider a fixed memory restriction and instead just compare the techniques within the same base abstraction.

For both 2-player and 3-player, for each of the three base abstractions, and for each player $i$, we build a base strategy, a dynamic expert strategy, an $S = \{i\}$ static expert strategy, and two $S = N$ static expert strategies. Recall choosing $S = \{i\}$ means that during computation of each static expert, we only fix player $i$'s action probabilities outside of the expert subtree, whereas $S = N$ means that we fix all players outside of the subtree. For 2-player Leduc, we use $r$, $cr$, $ccr$, and $cccr$-experts for both players. Thus, the base strategy plays until the first raise occurs, at which point an expert takes over for the remainder of the hand. As an exception, only one of our two $S = N$ static expert strategies, named "All," uses all four experts; the other, named "Pre-flop," just uses the $r$ and $cr$-experts. For 3-player Leduc, we use $r$, $cr$, $ccr$, $cccr$, $ccccr$, and $cccccr$-experts, except the "Pre-flop" static strategies use just the three experts $r$, $cr$, and $ccr$. The null abstraction is employed

**Table 1:** The size, earnings, and exploitability of the 2-player (2p) Leduc strategies in the JQ-K base abstraction, and the size and earnings of the 3-player (3p) strategies in the J-QK base abstraction. The sizes are measured in terms of the maximum number of information sets present within a single CFR computation. Earnings, as described in the text, and exploitability are in milli-antes per hand.

| Strategy (2p) | Size | Earns. | Exploit. | Strategy (3p) | Size | Earns. |
|---|---|---|---|---|---|---|
| Base | 132 | 24.73 | 496.31 | Base | 1890 | -68.46 |
| Dynamic | 444 | 45.75 | 159.84 | Dynamic | 6903 | 113.04 |
| Static.$S=\{i\}$ | 226 | 28.87 | 167.61 | Static.$S=\{i\}$ | 3017 | 96.14 |
| Static.$S=N$.All | 186 | 29.20 | 432.74 | Static.$S=N$.All | 2145 | 117.01 |
| Static.$S=N$.Pre-flop | 186 | 37.77 | 214.44 | Static.$S=N$.Pre-flop | 2145 | 119.73 |

on every expert subtree. Each run of CFR is stopped after 100 million iterations, which for 2-player yields strategies within a milli-ante of equilibrium in the abstract game.

Each strategy is evaluated against all combinations and orderings of opponent strategies where all strategies use different base abstractions, and the scores are averaged together. For example, for each of our 2-player strategy profiles $\sigma$ in the JQ-K base abstraction, we compute $1/2(u_1(\sigma_1, \sigma_2') + u_2(\sigma_1', \sigma_2))$, averaged over all profiles $\sigma'$ that use either the null or J-QK base abstraction. Leduc is a small enough game that the utilities can be computed exactly. A selection of these scores, along with 2-player exploitability values, are reported in Table 1.

Firstly, by increasing abstraction granularity, all of the JQ-K strategies employing experts earn more than the base strategy alone. Secondly, Dynamic and Static.$S=N$ earn more overall than Static.$S=\{i\}$, despite the 2-player Static.$S=N$ being more exploitable due to the opponent action assumptions. In fact, despite requiring much less memory to compute, Static.$S=N$ surprisingly earns more than Dynamic in 3-player Leduc. Finally, we see that only using two pre-flop static experts as opposed to all four reduces the number of dangerous assumptions to provide a stronger and less exploitable strategy. However, as expected, Dynamic and Static.$S=\{i\}$ are less exploitable.

**Hold'em.** Our Hold'em experiments enforce a fixed memory restriction per run of CFR, which we artificially set to 24 million information sets for 2-player and 162 million information sets for 3-player. We compute stitched strategies of each type using as many percentile $\mathbf{E}[HS^2]$ buckets as possible within the restriction. Our 2-player abstractions distribute buckets as close to uniformly as possible across the betting rounds while remembering buckets from previous rounds (known as "perfect recall"). Our 3-player abstractions are similar, except they use 169 pre-flop buckets that are forgotten on later rounds (known as "imperfect recall;" see [1] and [13] for more regarding CFR and imperfect recall).

For 2-player, our dynamic strategy has just an $r$-expert, our $S = \{i\}$ static strategy uses $r$, $cr$, $ccr$, and $cccr$-experts, and our $S = N$ static strategy employs $r$ and $cr$-experts. These choices were based on preliminary experiments to make the most effective use of the limited memory available for each stitching approach. Following Abou Risk and Szafron [1], our 3-player stitched strategies all have $f$, $rf$, $rrf$, and $rcf$-experts as these appear to be the most commonly reached 2-player scenarios [1, Table 4]. Our abstractions range quite dramatically in terms of number of buckets. For example, in 3-player, our dynamic strategy's base abstraction has just 8 river buckets with 7290 river buckets for each expert, whereas our static strategies have 16 river buckets in the base abstraction with up to 194,481 river buckets for the $S = N$ static $rcf$-expert abstraction. For reference, all of the 2-player base and experts are built from 720 million iterations of CFR, while we run CFR for 100 million and 5 billion iterations for the 3-player base and experts respectively.

We evaluate our 2-player strategies by playing 500,000 duplicate hands (players play both sides of the dealt cards) of poker between each pair of strategies. In addition to our base and stitched strategies, we also included a base strategy called "Base.797M" in an abstraction with over 797 million information sets that we expected to beat all of the strategies we were evaluating. Furthermore, using a specialized best response tool [8], we computed the exploitability of our 2-player strategies. For 3-player, we play 500,000 triplicate hands (each set of dealt cards played 6 times, one for each of the player seatings) between each combination of 3 strategies. We also included two other strategies: "ACPC-09," the 2009 ACPC 3-player event winner that did not use experts (Abou Risk and Szafron [1] call it "IR16"), and "ACPC-10," a static expert strategy that won a 3-player event at the 2010 ACPC and is outlined at the end of this section. The results are provided in Table 2.

**Table 2:** Earnings and 95% confidence intervals over 500,000 duplicate hands of 2-player Hold'em per pairing, and over 500,000 triplicate hands of 3-player Hold'em per combination. The exploitability of the 2-player strategies is also provided. All values are in milli-big-blinds per hand.

| Strategy (2p) | Earnings | Exploitability | Strategy (3p) | Earnings |
|---|---|---|---|---|
| Base | $-10.47 \pm 1.99$ | 310.04 | Base | $-6.09 \pm 0.71$ |
| Dynamic | $-4.43 \pm 1.98$ | 307.76 | Dynamic | $-4.91 \pm 0.75$ |
| Static.$S=\{i\}$ | $-13.13 \pm 2.00$ | 301.00 | Static.$S=\{i\}$ | $-5.20 \pm 0.70$ |
| Static.$S=N$ | $-4.57 \pm 1.95$ | 288.82 | Static.$S=N$ | $3.06 \pm 0.70$ |
| Base.797M | $32.59 \pm 2.14$ | 135.43 | ACPC-09 | $-14.15 \pm 0.89$ |
| | | | ACPC-10 | $27.29 \pm 0.86$ |

Firstly, in 2-player, we see that Static.$S=N$ and Dynamic outperform Static.$S=\{i\}$ considerably, agreeing with the previous Leduc results. In fact, the Static.$S=\{i\}$ fails to even improve upon the base strategy. For 3-player, Static.$S=N$ is noticeably ahead of both Dynamic and Static.$S=\{i\}$ as it is the only strategy, aside from ACPC-10, to win money. By forcing one player to fold, the static experts with $S = N$ essentially reduce the size of the game tree from a 3-player to a 2-player game, allowing many more buckets to be used. This result indicates that at least for poker, the gains in abstraction bucketing outweigh the risks of forced action assumptions and lack of cohesion between the base strategy and the experts. Furthermore, Static.$S=N$ is slightly less exploitable in 2-player than the base strategy and the other two stitched strategies. While there are one and two opponent static actions assumed by the $r$ and $cr$-experts respectively, trading these few assumptions for an increase in abstraction granularity is beneficial. In summary, static experts with $S = N$ are preferred to both dynamic and static experts with $S = \{i\}$ in the experiments we ran.

An additional validation of the quality of the static expert approach was provided by the 2010 ACPC. The winning entries in both 3-player events employed static experts with $S = N$. The base strategy, computed from 70 million iterations of CFR, used $169, 900, 100$, and 25 buckets on each of the respective rounds. Four experts were used, $f, rf, rrf$, and $rcf$, computed from 10 billion iterations of CFR, each containing 169, 60,000, 180,000, and 26,160 buckets on the respective rounds. In addition, clustering techniques on strength distribution were used instead of percentile bucketing. Two strategies were created, where one was trained to play slightly more aggressively for the total bankroll event. Each version finished in first place in its respective competition.

## 7 Conclusions

We discussed two strategy stitching techniques for extensive games, including static experts that generalize strategy grafting and some previous techniques used in poker. Despite the accompanying potential dangers and lack of cohesion, we have shown static experts with $S = N$ outperform the dynamic and static experts with $S = \{i\}$ that we considered, especially when memory limitations are present. However, additional static experts with several forced actions can lead to a more exploitable strategy. Static experts with $S = N$ is currently our preferred method for creating multiplayer poker strategies and would be our first option for playing other large extensive games.

Future work includes finding a way to create more cohesion between the base strategy and static experts. One possibility is to rebuild the base strategy after the experts have been created so that the base strategy's play is more unified with the experts. In addition, we have yet to experiment with 3-player "hybrid" static experts where $|S| = 2$. Finally, there are many ways to combine the stitching techniques described in this paper. One possibility is to use a dynamic expert strategy as a base strategy of a static expert strategy. In addition, static experts could themselves be dynamic expert strategies for the appropriate subtrees. Such combinations may produce even stronger strategies than those produced in this paper.

## Acknowledgments

We would like to thank Westgrid and Compute Canada for their computing resources that were used during this work. We would also like to thank the members of the Computer Poker Research Group at the University of Alberta for their helpful pointers throughout this project. This research was funded by NSERC and Alberta Ingenuity, now part of Alberta Innovates - Technology Futures.

## Footnotes

[1]http://www.computerpokercompetition.org

[2]http://en.wikipedia.org/wiki/Texas_hold_'em

# References

[1] N. Abou Risk and D. Szafron. Using counterfactual regret minimization to create competitive multiplayer poker agents. In *AAMAS*, pages 159–166, 2010.

[2] D. Billings, N. Burch, A. Davidson, R. Holte, J. Schaeffer, T. Schauenberg, and D. Szafron. Approximating game-theoretic optimal strategies for full-scale poker. In *IJCAI*, pages 661–668, 2003.

[3] S. Ganzfried and T. Sandholm. Computing an approximate jam/fold equilibrium for 3-agent no-limit Texas Hold'em tournaments. In *AAMAS*, 2008.

[4] S. Ganzfried and T. Sandholm. Computing equilibria in multiplayer stochastic games of imperfect information. In *IJCAI*, 2009.

[5] A. Gilpin and T. Sandholm. Better automated abstraction techniques for imperfect information games, with application to Texas Hold'em poker. In *AAMAS*, 2007.

[6] A. Gilpin, T. Sandholm, and T.B. Sørensen. Potential-aware automated abstraction of sequential games, and holistic equilibrium analysis of Texas Hold'em poker. In *AAAI*, 2007.

[7] M. Johanson. Robust strategies and counter-strategies: Building a champion level computer poker player. Master's thesis, University of Alberta, 2007.

[8] M. Johanson, K. Waugh, M. Bowling, and M. Zinkevich. Accelerating best response calculation in large extensive games. In *IJCAI*, 2011. To appear.

[9] M. Osborne and A. Rubenstein. *A Course in Game Theory*. The MIT Press, Cambridge, Massachusetts, 1994.

[10] F. Southey, M. Bowling, B. Larson, C. Piccione, N. Burch, D. Billings, and C. Rayner. Bayes' bluff: Opponent modelling in poker. In *UAI*, pages 550–558, 2005.

[11] K. Waugh, M. Bowling, and N. Bard. Strategy grafting in extensive games. In *NIPS-22*, pages 2026–2034, 2009.

[12] K. Waugh, D. Schnizlein, M. Bowling, and D. Szafron. Abstraction pathologies in extensive games. In *SARA*, pages 781–788, 2009.

[13] Kevin Waugh, Martin Zinkevich, Michael Johanson, Morgan Kan, David Schnizlein, and Michael Bowling. A practical use of imperfect recall. In *SARA*, pages 175–182, 2009.

[14] M. Zinkevich, M. Johanson, M. Bowling, and C. Piccione. Regret minimization in games with incomplete information. In *NIPS-20*, pages 905–912, 2008.

